# A Computational Model for Cursive Handwriting Based on the Minimization Principle

**Yasuhiro Wada** *     **Yasuharu Koike**

**Eric Vatikiotis-Bateson**     **Mitsuo Kawato**

ATR Human Information Processing Research Laboratories
2-2 Hikaridai, Seika-cho, Soraku-gun, Kyoto 619-02, Japan

## ABSTRACT

We propose a trajectory planning and control theory for continuous movements such as connected cursive handwriting and continuous natural speech. Its hardware is based on our previously proposed forward-inverse-relaxation neural network (Wada & Kawato, 1993). Computationally, its optimization principle is the minimum torque-change criterion. Regarding the representation level, hard constraints satisfied by a trajectory are represented as a set of via-points extracted from a handwritten character. Accordingly, we propose a via-point estimation algorithm that estimates via-points by repeating the trajectory formation of a character and the via-point extraction from the character. In experiments, good quantitative agreement is found between human handwriting data and the trajectories generated by the theory. Finally, we propose a recognition schema based on the movement generation. We show a result in which the recognition schema is applied to the handwritten character recognition and can be extended to the phoneme timing estimation of natural speech.

## 1 INTRODUCTION

In reaching movements, trajectory formation is an ill-posed problem because the hand can move along an infinite number of possible trajectories from the starting to the target point. However, humans move an arm between two targets along consistent one of an

             Makuhari Techno Garden, 1-3 Nakase, Mihama-ku, Chiba 261, Japan

infinite number of trajectories. Therefore, the brain should be able to compute a unique solution by imposing an appropriate criterion to the ill-posed problem. Especially, a smoothness performance index was intensively studied in this context.

Flash & Hogan (1985) proposed a mathematical model, the minimum-jerk model. Their model is based on the kinematics of movement, independent of the dynamics of the musculoskeletal system. On the other hand, based on the idea that the objective function must be related to dynamics, Uno, Kawato & Suzuki (1989) proposed the minimum torque-change criterion which accounts for the desired trajectory determination. The criterion is based on the theory that the trajectory of the human arm is determined so as to minimize the time integral of the square of the rate of torque change. They proposed the following quadratic measure of performance. Where $\tau^j$ is the torque generated by the $j$-th actuator of $M$ actuators, and $t_f$ is the movement time.

$$C_T = \int_0^{t_f} \sum_{j=1}^{M} \left( \frac{d\tau^j}{dt} \right)^2 dt \qquad (1)$$

Handwriting production is an attractive subject in human motor control studies. In cursive handwriting, a symbol must be transformed into a motor command stream. This transformation process raises several questions. How can the central nervous system (CNS) represent a character symbol for producing a handwritten letter? By what principle can motor planning be made or a motor command be produced? In this paper we propose a handwriting model whose computational theory and representation are the same as the model in reaching movements. Our proposed computational model for cursive handwriting is assumed to generate a trajectory that passes through many via-points. The computational theory is based on the minimum torque-change criterion, and a representation of a character is assumed to be expressed as a set of via-points extracted from a handwritten character. In reaching movement, the boundary condition is given by the visual information, such as the location of a cup, and the trajectory formation is based on the minimum torque-change criterion, which is completely the same as the model of handwriting (Fig. 1). However, it is quite difficult to determine the via-points in order to reproduce a cursive handwritten character. We propose an algorithm that can determine the via-points of the handwritten character, based only on the same minimization principle and which does not use any other *ad hoc* information such as zero-crossing velocity (Hollerbach, 1981).

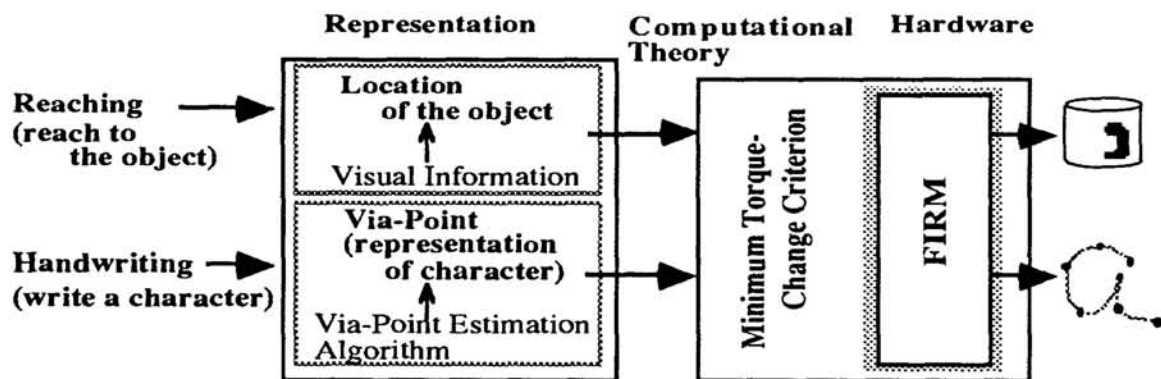

Figure 1: A handwriting model.

## 2   PREVIOUS WORK ON THE HANDWRITING MODEL

Several handwriting models (Hollerbach, 1981; Morasso & Mussa-Ivaldi, 1982; Edleman & Flash, 1987) have been proposed. Hollerbach proposed a handwriting model based on oscillation theory. The model basically used a vertical oscillator and a horizontal oscillator. Morasso & Mussa-Ivaldi proposed a trajectory formation model using a spline function, and realized a handwritten character using the formation model.

Edleman & Flash (1987) proposed a handwriting model based on snap (fourth derivative of position) minimization. The representation of a character was four basic strokes and a handwritten character was regenerated by a combination of several strokes. However, their model was different from their theory for reaching movement. Flash & Hogan (1985) have proposed the minimum jerk criterion in the reaching movement.

## 3   A HANDWRITING MODEL

### 3.1   Trajectory formation neural network:
### Forward-Inverse Relaxation Model (FIRM)

First, we explain the trajectory formation neural network. Because the dynamics of the human arm are nonlinear, finding a unique trajectory based on the minimum torque-change criterion is a nonlinear optimization problem. Moreover, it is rather difficult. There are several criticisms of previous proposed neural networks based on the minimum torque-change criterion: (1) their spatial representation of time, (2) back propagation is essential, and (3) much time is required. Therefore, we have proposed a new neural network, FIRM(Forward-Inverse Relaxation Model) for trajectory formation (Wada & Kawato, 1993). This network can be implemented as a biologically plausible neural network and resolve the above criticisms.

### 3.2   Via-point estimation model

Edelman & Flash (1987) have pointed out the difficulty of finding the via-points in a handwritten character. They have argued two points: (1) the number of via-points, (2) a reason for the choice of every via-point locus. It is clear in approximation theory that a character can be regenerated perfectly if the number of extracted via-points is large. Appropriate via-points can not be assigned according to a regular sampling rule if the sample duration is constant and long. Therefore, there is an infinite number of combinations of numbers and via-point positions in the problem of extracting via-points from a given trajectory, and a unique solution can not be found if a trajectory reformation theory is not identified. That is, it is an ill-posed problem.

The algorithm for assigning the via-points finds the via-points by iteratively activating both the trajectory formation module (FIRM) and the via-point extraction module (Fig. 2). The trajectory formation module generates a trajectory based on the minimum torque-change criterion using the via-points which are extracted by the via-point extraction module. The via-point extraction module assigns the via-points so as to minimize the square error between the given trajectory and the trajectory generated by the trajectory formation module. The via-point extraction algorithm will stop when the error between the given trajectory and the trajectory generated from the extracted via-points reaches a threshold.

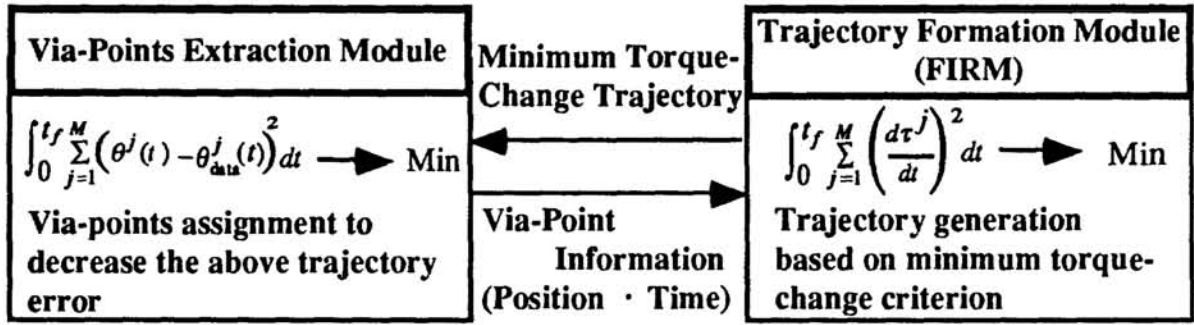

Figure 2: Via-point estimation model. $\theta_{data}^j(t)$ is the given trajectory of the j-th joint angle and $\theta^j(t)$ represents the generated trajectory.

### 3.2.1 Algorithm of via-point extraction

There are a via-point extraction procedure and a trajectory production procedure in the via-point extraction module, and they are iteratively computed. Trajectory production in the module is based on the minimum-jerk model (Flash & Hogan 1985) on a joint angle space, which is equivalent to the minimum torque-change model when arm dynamics are approximated as in the following dynamic equation:

$$\tau^j = I^j \ddot{\theta}^j \qquad (j = 1, \cdots, M) \qquad (2)$$

where $I^j$ and $\ddot{\theta}^j$ are the inertia of the link and the acceleration of the j-th joint angle, respectively.

The algorithm for via-point extraction is illustrated in Fig. 3. The procedural sequence is as follows:

(**Step 1**) A trajectory between a starting point and a final point is generated by using the minimum torque-change principle of the linear dynamics model.

(**Step 2**) The point with the maximum square error value between the given trajectory and the generated trajectory is selected as a via-point candidate.

(**Step 3**) If the maximum value of the square error is less than the preassigned threshold, the procedure described above is finished. If the maximum value of the square error is greater than the threshold, the via-point candidate is assigned as via-point $i$ and a trajectory is generated from the starting point through the via-point $i$ to the final point. This generated trajectory is added to the trajectory that has already been generated. The time of the start point of the generated trajectory is a via-point located just before the assigned via-point $i$, and the time of the final point of the generated trajectory is a via-point located just after the assigned via-point $i$. The position error of the start point and the final point equal 0, since the compensation for the error has already been made. Thus, the boundary conditions of the generated trajectory at the start and final point become 0. The velocity and acceleration constraints at the start and final point are set to 0.

(**Step 4**) By repeating Steps 2 and 3, a set of via-points is found.

The j-th actuator velocity constraint $\dot{\theta}_{via}^j$ and acceleration constraint $\ddot{\theta}_{via}^j$ at the via-point $i$ are set by minimizing the following equation.

$$J(\dot{\theta}_{via}^j, \ddot{\theta}_{via}^j) = I^{j2} \left\{ \int_{t_0^i}^{t_{via}^i} \left( \dddot{\theta}^j \right)^2 dt + \int_{t_{via}^i}^{t_f^i} \left( \dddot{\theta}^j \right)^2 dt \right\} \rightarrow Min \qquad (3)$$

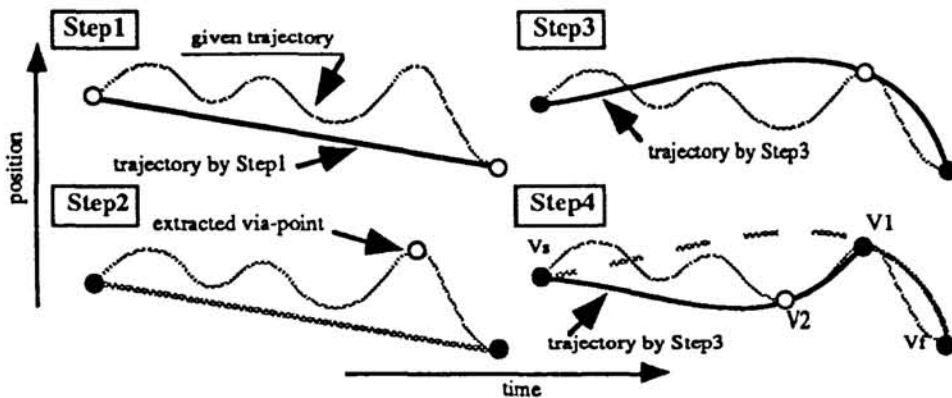

Figure 3: An algorithm for extracting via-points.

Finally, the via-points are fed to the FIRM, and the minimum torque change trajectory is produced. This trajectory and the given trajectory are then compared again. If the value of the square error does not reach the threshold, the procedure above is repeated.

It can be mathematically shown that a given trajectory is perfectly approximated with this method (completeness), and furthermore that the number of extracted via-points for a threshold is the minimum (optimality). (Wada & Kawato, 1994)

## 4    PERFORMANCE OF THE VIA-POINT ESTIMATION MODEL

### 4.1    Performance of single via-point movement

First, we examine the performance of our proposed via-point estimation model. A result of via-point estimation in a movement with a via-point is shown in Fig 4. Two movements (T3-P1-T5 and T3-P2-T5) are examined. The white circle and the solid lines show the target points and measured trajectories, respectively. P1 and P2 show target via-points. The black circle shows the via-points estimated by the algorithm. The estimated via-points were close to the target via-points. Thus, our proposed via-point estimation algorithm can find a via-point on the given trajectory.

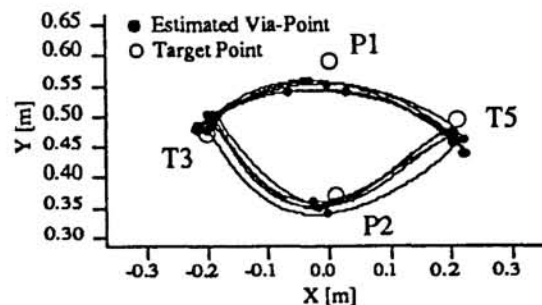

Figure 4: A result of via-point estimation in a movement with a via-point.

### 4.2    Performance of the handwriting model

Fig. 5 shows the case of cursive connected handwritten characters. The handwriting model can generate trajectories and velocity curves of cursive handwritten characters that are almost identical to human data. The estimated via-points are classified into two groups. The via-points in one group are extracted near the minimum points of the

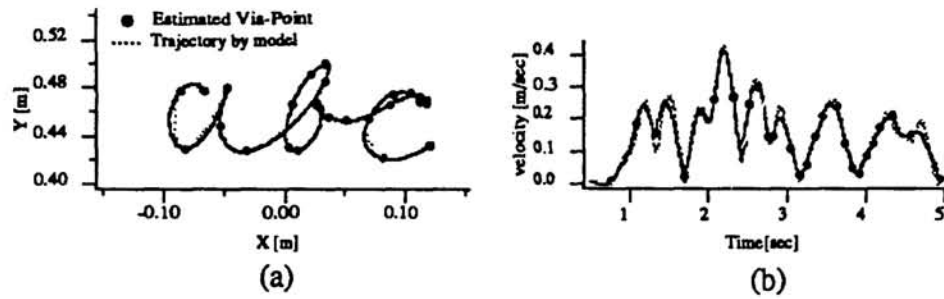

Figure 5: Estimated via-points in cursive handwriting. (a) and (b) show the trajectory and tangential velocity profile, respectively. The via-point estimation algorithm extracts a via-point (segmentation point) between characters.

velocity profile. The via-points of the other group are assigned to positions that are independent of the above points. Generally, the minimums of the velocity are considered to be the feature points of the movement. However, we confirmed that a given trajectory can not be reproduced by using only the first group of via-points. This finding shows that the second group of via-points is important. Our proposed algorithm based on the minimization principle can estimate points that can not be selected by any kinematic criterion. Furthermore, it is important in handwritten character recognition that the via-point estimation algorithm extracts via-points between characters, that is, their segmentation points.

## 5    FROM FORMATION TO RECOGNITION

### 5.1  A recognition model

Next, we propose a recognition system using the trajectory formation model and the via-point estimation model. There are several reports in the literature of psychology which suggest that the formation process is related to the recognition process. (Liberman & Mattingly, 1985; Freyd, 1983)

Here, we present a pattern recognition model that strongly depends on the handwriting model and the via-point estimation model (Fig.6). (1) The features of the handwritten character are extracted by the via-point estimation algorithm. (2) Some of via-points are segmented and normalized in space and time. Then, (3) a trajectory is regenerated by using the normalized via-points. (4) A symbol is identified by comparing the regenerated trajectory with the template trajectory.

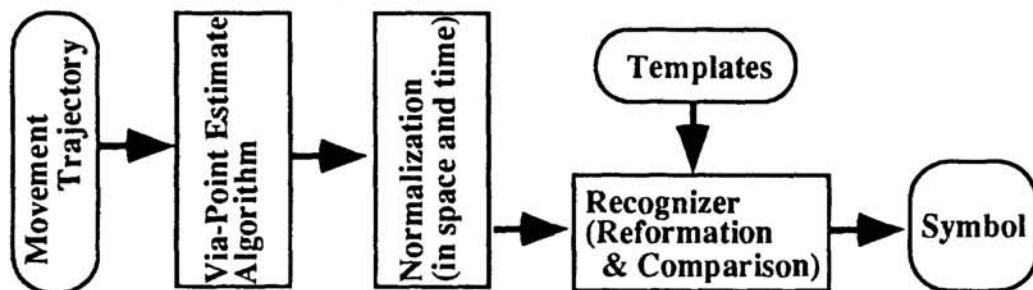

Figure 6: Movement pattern recognition using extracted via-points obtained through movement pattern generator

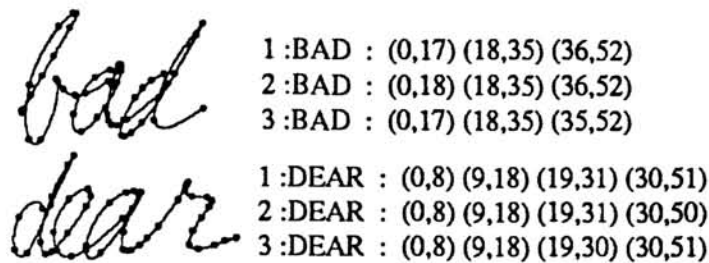

1 :BAD : (0,17) (18,35) (36,52)
2 :BAD : (0,18) (18,35) (36,52)
3 :BAD : (0,17) (18,35) (35,52)

1 :DEAR : (0,8) (9,18) (19,31) (30,51)
2 :DEAR : (0,8) (9,18) (19,31) (30,50)
3 :DEAR : (0,8) (9,18) (19,30) (30,51)

Figure 7: Results of character recognition

## 5.2 Performance of the character recognition model

Fig. 7 shows a result of character recognition. The right-hand side shows the recognition results for the left-hand side. The best three candidates for recognition are listed. Numerals in parentheses show the number of starting via-points and the final via-point for the recognized character.

## 5.3 Performance of the estimation of timing of phonemes in real speech

Fig. 8 shows the acoustic waveform, the spectrogram, and the articulation movement when the sentence " Sam sat on top of the potato cooker..." is spoken. The phonemes are identified, and the vertical lines denote phoneme midpoints. White circles show the via-points estimated by our proposed algorithm. Rather good agreement is found between the estimated via-points and the phonemes.

From this experiment, we can point out two important possibilities for the estimation model of phoneme timing. The first possibility concerns speech recognition, and the second concerns speech data compression. It seems possible to extend the via-point estimation algorithm to speech recognition if a mapping from acoustic to articulator motion is identified (Shirai & Kobayashi, 1991, Papcun et al., 1992). Furthermore, with training of a forward mapping from articulator motion to acoustic data (Hirayama et al., 1993), the via-point estimation model can be used for speech data compression.

## 6  SUMMARY

We have proposed a new handwriting model. In experiments, good qualitative and quantitative agreement is found between human handwriting data and the trajectories generated by the model. Our model is unique in that the same optimization principle and hard constraints used for reaching are also used for cursive handwriting. Also, as opposed to previous handwriting models, determination of via-points is based on the optimization principle and does not use a priori knowledge.

We have demonstrated two areas of recognition, connected cursive handwritten character recognition and the estimation of phoneme timing. We incorporated the formation model into the recognition model and realized the recognition model suggested by Freyd (1983) and Liberman and Mattingly(1985). The most important point shown by the models is that the human recognition process can be realized by specifying the human formation process.

## Footnotes

* Present Address: Systems Lab., Kawasaki Steel Corporation,

## REFERENCES

S. Edelman & T. Flash (1987) A Model of Handwriting. *Biol. Cybern.* , **57**, 25-36.

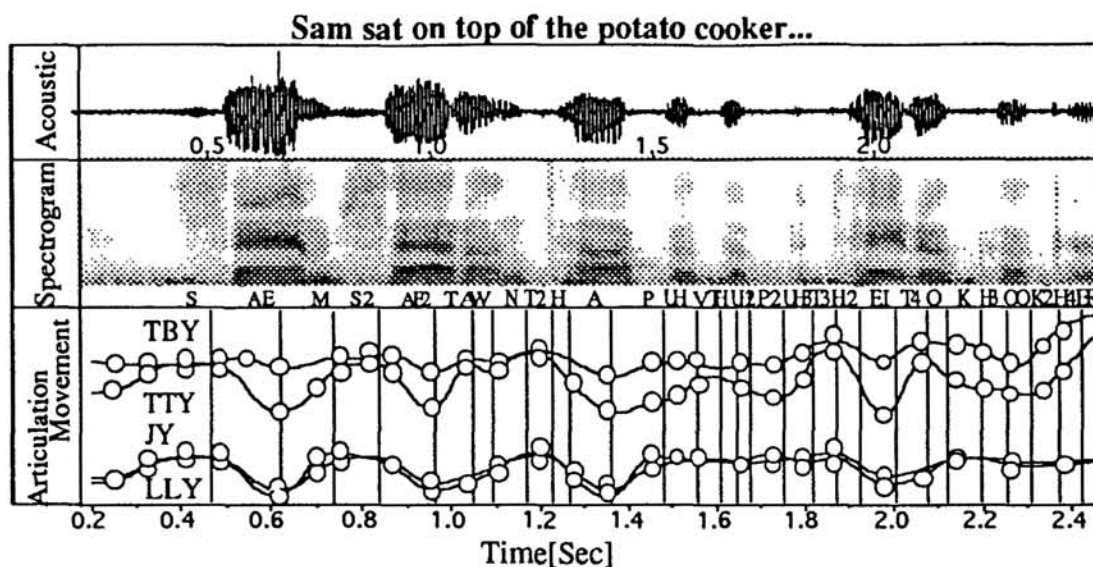

Figure 8: Estimation result of phoneme time. Temporal acoustics and vertical positions of the tongue blade (TBY),tongue tip (TTY), jaw (JY), and lower lip (LLY) are shown with overlaid via-point trajectories. Vertical lines correspond to acoustic segment centers; O denotes via-points.

T. Flash, & N. Hogan (1985) The coordination of arm movements; An experimentally confirmed mathematical model. *Journal of Neuroscience*, 5, 1688-1703.

J. J. Freyd (1983) Representing the dynamics of a static form. *Memory & Cognition*, 11, 342-346.

M. Hirayama, E. Vatikiotis-Bateson, K. Honda, Y. Koike, & M. Kawato (1993) Physiologically based speech synthesis. In Giles, C. L., Hanson, S. J., and Cowan, J. D. (eds) *Advances in Neural Information Processing Systems 5*, 658-665. San Mateo, CA: Morgan Kaufmann Publishers.

J. M. Hollerbach (1981) An oscillation theory of handwriting. *Biol. Cybern.*, 39,139-156.

A. M. Liberman & I. G. Mattingly (1985) The motor theory of speech perception revised. *Cognition*, 21, 1-36.

P. Morasso, & F. A. Mussa-Ivaldi (1982) Trajectory formation and handwriting: A computational model. *Biol. Cybern.*, 45, 131-142.

J. Papcun, J. Hochberg, T. R. Thomas, T. Laroche, J. Zacks, & S. Levy (1992) Inferring articulation and recognition gestures from acoustics with a neural network trained on x-ray microbeam data. *Journal of Acoustical Society of America*, 92 (2) **Pt. 1.**

K. Shirai, & T. Kobayashi (1991) Estimation of articulatory motion using neural networks. *Journal of Phonetics*, 19, 379-385.

Y. Uno, M. Kawato, & R. Suzuki (1989) Formation and control of optimal trajectory in human arm movement - minimum torque-change model. *Biol. Cybern.* 61, 89-101.

Y. Wada, & M. Kawato (1993) A neural network model for arm trajectory formation using forward and inverse dynamics models. *Neural Networks*, 6(7),919-932.

Y. Wada, & M. Kawato (1994) Long version of this paper, in preparation.
